# Contour Organisation with the EM Algorithm

**J. A. F. Leite and E. R. Hancock**
Department of Computer Science
University of York, York, Y01 5DD, UK.

## Abstract

This paper describes how the early visual process of contour organisation can be realised using the EM algorithm. The underlying computational representation is based on fine spline coverings. According to our EM approach the adjustment of spline parameters draws on an iterative weighted least-squares fitting process. The expectation step of our EM procedure computes the likelihood of the data using a mixture model defined over the set of spline coverings. These splines are limited in their spatial extent using Gaussian windowing functions. The maximisation of the likelihood leads to a set of linear equations in the spline parameters which solve the weighted least squares problem. We evaluate the technique on the localisation of road structures in aerial infra-red images.

## 1 Introduction

Dempster, Laird and Rubin's EM (expectation and maximisation) [1] algorithm was originally introduced as a means of finding maximum likelihood solutions to problems posed in terms of incomplete data. The basic idea underlying the algorithm is to iterate between the expectation and maximisation modes until convergence is reached. Expectation involves computing *a posteriori* model probabilities using a mixture density specified in terms of a series of model parameters. In the maximisation phase, the model parameters are recomputed to maximise the expected value of the incomplete data likelihood. In fact, when viewed from this perspective, the updating of *a posteriori* probabilities in the expectation phase would appear to have much in common with the probabilistic relaxation process extensively exploited in low and intermediate level vision [9, 2]. Maximisation of the incomplete

data likelihood is reminiscent of robust estimation where outlier reject is employed in the iterative re-computation of model parameters [7].

It is these observations that motivate the study reported in this paper. We are interested in the organisation of the output of local feature enhancement operators into meaningful global contour structures [13, 2]. Despite providing one of the classical applications of relaxation labelling in low-level vision [9], successful solutions to the iterative curve reinforcement problem have proved to be surprisingly elusive [8, 12, 2]. Recently, two contrasting ideas have offered practical relaxation operators. Zucker *et al* [13] have sought biologically plausible operators which draw on the idea of computing a global curve organisation potential and locating consistent structure using a form of local snake dynamics [11]. In essence this biologically inspired model delivers a fine arrangement of local splines that minimise the curve organisation potential. Hancock and Kittler [2], on the other hand, appealed to a more information theoretic motivation [4]. In an attempt to overcome some of the well documented limitations of the original Rosenfeld, Hummel and Zucker relaxation operator [9] they have developed a Bayesian framework for relaxation labelling [4]. Of particular significance for the low-level curve enhancement problem is the underlying statistical framework which makes a clear-cut distinction between the roles of uncertain image data and prior knowledge of contour structure. This framework has allowed the output of local image operators to be represented in terms of Gaussian measurement densities, while curve structure is represented by a dictionary of consistent contour structures [2].

While both the fine-spline coverings of Zucker [13] and the dictionary-based relaxation operator of Hancock and Kittler [2] have delivered practical solutions to the curve reinforcement problem, they each suffer a number of shortcomings. For instance, although the fine spline operator can achieve quasi-global curve organisation, it is based on an essentially *ad hoc* local compatibility model. While being more information theoretic, the dictionary-based relaxation operator is limited by virtue of the fact that in most practical applications the dictionary can only realistically be evaluated over at most a 3x3 pixel neighbourhood. Our aim in this paper is to bridge the methodological divide between the biologically inspired fine-spline operator and the statistical framework of dictionary-based relaxation. We develop an iterative spline fitting process using the EM algorithm of Dempster *et al* [1]. In doing this we retain the statistical framework for representing filter responses that has been used to great effect in the initialisation of dictionary-based relaxation. However, we overcome the limited contour representation of the dictionary by drawing on local cubic splines.

## 2 Prerequisites

The practical goal in this paper is the detection of line-features which manifest themselves as intensity ridges of variable width in raw image data. Each pixel is characterised by a vector of measurements, $z_i$ where $i$ is the pixel-index. This measurement vector is computed by applying a battery of line-detection filters of various widths and orientations to the raw image. Suppose that the image data is indexed by the pixel-set $I$. Associated with each image pixel is a cubic spline parameterisation which represents the best-fit contour that couples it to adjacent feature pixels. The spline is represented by a vector of parameters denoted by

$\underline{q}_i = (q_i^0, q_i^1, q_i^2, q_i^3)^T$. Let $(x_i, y_i)$ represent the position co-ordinates of the pixel indexed $i$. The spline variable, $s_{i,j} = x_i - x_j$ associated with the contour connecting the pixel indexed $j$ is the horizontal displacement between the pixels indexed $i$ and $j$. We can write the cubic spline as an inner product $F(s_{i,j}, \underline{q}_i) = \underline{q}_i^T . \underline{S}_{i,j}$ where $\underline{S}_{i,j} = (1, s_{i,j}, s_{i,j}^2, s_{i,j}^3)^T$. Central to our EM algorithm will be the comparison of the predicted vertical spline displacement with its measured value $r_{i,j} = y_i - y_j$.

In order to initialise the EM algorithm, we require a set of initial spline probabilities which we denote by $\pi(\underline{q}_i^{(0)})$. Here we use the multi-channel combination model recently reported by Leite and Hancock [5] to compute an initial multi-scale line-feature probability. Accordingly, if $\Sigma$ is the variance-covariance matrix for the components of the filter bank, then

$$\pi(\underline{q}_i^{(0)}) = 1 - \exp\left[-\frac{1}{2}\underline{z}_i^T \Sigma^{-1} \underline{z}_i\right] \qquad (1)$$

The remainder of this paper outlines how these initial probabilities are iteratively refined using the EM algorithm. Because space is limited we only provide an algorithm sketch. Essential algorithm details such as the estimation of spline orientation and the local receptive gating of the spline probabilities are omitted for clarity. Full details can be found in a forthcoming journal article [6].

## 3   Expectation

Our basic model of the spline organisation process is as follows. Associated with each image pixel is a spline parameterisation. Key to our philosophy of exploiting a mixture model to describe the global contour structure of the image is the idea that the pixel indexed $i$ can associate to each of the putative splines residing in a local Gaussian window $N_i$. We commence by developing a mixture model for the conditional probability density for the filter response $\underline{z}_i$ given the current global spline description. If $\Phi^{(n)} = \{\underline{q}_i^{(n)}, \forall i \in I\}$ is the global spline description at iteration $n$ of the EM process, then we can expand the mixture distribution over a set of putative splines that may associate with the image pixel indexed $i$

$$p(\underline{z}_i | \Phi^{(n)}) = \sum_{j \in N_i} p(\underline{z}_i | \underline{q}_j^{(n)}) \pi(\underline{q}_j^{(n)}) \qquad (2)$$

The components of the above mixture density are the conditional measurement densities $p(\underline{z}_i | \underline{q}_j^{(n)})$ and the spline mixing proportions $\pi(\underline{q}_j^{(n)})$. The conditional measurement densities represent the likelihood that the datum $\underline{z}_i$ originates from the spline centred on pixel $j$. The mixing proportions, on the other hand, represent the fractional contribution to the data arising from the $j$th parameter vector i.e. $\underline{q}_j^{(n)}$. Since we are interested in the maximum likelihood estimation of spline parameters, we turn our attention to the likelihood of the raw data, i.e.

$$p(\underline{z}_i, \forall i \in I | \Phi^{(n)}) = \prod_{i \in I} p(\underline{z}_i | \Phi^{(n)}) \qquad (3)$$

The expectation step of the EM algorithm is aimed at estimating the log-likelihood using the parameters of the mixture distribution. In other words, we need to average the likelihood over the space of potential pixel-spline assignments. In fact,

it was Dempster, Laird and Rubin [1] who observed that maximising the weighted log-likelihood was equivalent to maximising the conditional expectation of the likelihood for a new parameter set given an old parameter set. For our spline fitting problem, maximisation of the expectation of the conditional likelihood is equivalent to maximising the weighted log-likelihood function

$$Q(\Phi^{(n+1)}|\Phi^{(n)}) = \sum_{i \in I} \sum_{j \in N_i} P(\underline{q}_j^{(n)}|\underline{z}_i) \ln p(\underline{z}_i|\underline{q}_j^{(n+1)}) \qquad (4)$$

The *a posteriori* probabilities $P(\underline{q}_j^{(n)}|\underline{z}_i)$ may be computed from the corresponding components of the mixture density $p(\underline{z}_i|q_j^{(n)})$ using the Bayes formula

$$P(\underline{q}_j^{(n)}|\underline{z}_i) = \frac{p(\underline{z}_i|\underline{q}_j^{(n)})\pi(\underline{q}_j^{(n)})}{\sum_{k \in N_i} p(\underline{z}_i|\underline{q}_k^{(n)})\pi(\underline{q}_k^{(n)})} \qquad (5)$$

For notational convenience, and to make the weighting role of the *a posteriori* probabilities explicit we use the shorthand $w_{i,j}^{(n)} = P(\underline{q}_j^{(n)}|\underline{z}_i)$. Once updated parameter estimates $\underline{q}_i^{(n)}$ become available through the maximisation of this criterion, improved estimates of the mixture components may be obtained by substitution into equation (6). The updated mixing proportions, $\pi(\underline{q}_i^{(n+1)})$, required to determine the new weights $w_{i,j}^{(n)}$ are computed from the newly available density components using the following estimator

$$\pi(\underline{q}_i^{(n+1)}) = \sum_{j \in N_i} \frac{p(\underline{z}_j|\underline{q}_i^{(n)})\pi(\underline{q}_i^{(n)})}{\sum_{k \in I} p(\underline{z}_j|\underline{q}_k^{(n)})\pi(\underline{q}_k^{(n)})} \qquad (6)$$

In order to proceed with the development of a spline fitting process we require a model for the mixture components, i.e. $p(\underline{z}_i|\underline{q}_j^{(n)})$. Here we assume that the required model can be specified in terms of Gaussian distribution functions. In other words, we confine our attention to Gaussian mixtures. The physical variable of these distributions is the squared error residual for the position prediction of the $i$th datum delivered by the $j$th spline. Accordingly we write

$$p(\underline{z}_i|\underline{q}_j^{(n)}) = \sqrt{\frac{\beta}{2\pi}} \exp\left[-\beta\left(r_{i,j} - F(s_{i,j}, \underline{q}_j^{(n)})\right)^2\right] \qquad (7)$$

where $\beta$ is the inverse variance of the fit residuals. Rather than estimating $\beta$, we use it in the spirit of a control variable to regulate the effect of fit residuals.

Equations (5), (6) and (11) therefore specify a recursive procedure that iterates the weighted residuals to compute a new mixing proportions based on the quality of the spline fit.

## 4  Maximisation

The maximisation step aims to optimize the quantity $Q(\Phi^{(n+1)}|\Phi^{(n)})$ with respect to the spline parameters. Formally this corresponds to finding the set of spline parameters which satisfy the condition

$$\Phi^{(n+1)} = \arg\max_{\Phi} Q(\Phi|\Phi^{(n)}) \qquad (8)$$

We find a local approximation to this condition by solving the following set of linear equations

$$\frac{\partial Q(\Phi^{(n+1)}|\Phi^{(n)})}{\partial (q_i^k)^{(n+1)}} = 0 \tag{9}$$

for each spline parameter $(q_i^k)^{(n+1)}$ in turn, i.e. for k=0,1,2,3. Recovery of the splines is most conveniently expressed in terms of the following matrix equation for the components of the parameter-vector $\underline{q}_i^{(n)}$

$$\underline{q}_i^{(n+1)} = (A_i^{(n)})^{-1}\underline{X}_i^{(n)} \tag{10}$$

The elements of the vector $X^{(n)}$ are weighted cross-moments between the parallel and perpendicular spline distances in the Gaussian window, i.e.

$$\underline{X}_i^{(n)} = \begin{pmatrix} \sum_{j \in N_i} w_{i,j}^{(n)} r_{i,j} \\ \sum_{j \in N_i} w_{i,j}^{(n)} r_{i,j} s_{i,j} \\ \sum_{j \in N_i} w_{i,j}^{(n)} r_{i,j} s_{i,j}^2 \\ \sum_{j \in N_i} w_{i,j}^{(n)} r_{i,j} s_{i,j}^3 \end{pmatrix} \tag{11}$$

The elements of the matrix $A_i^{(n)}$, on the other hand, are weighted moments computed purely in terms of the parallel distance $s_{i,j}$. If $k$ and $l$ are the row and column indices, then the $(k, l)$th element of the matrix $A_i^{(n)}$ is

$$[A_i^{(n)}]_{k,l} = \sum_{j \in N_i} w_{i,j}^{(n)} s_{i,j}^{k+l-2} \tag{12}$$

## 5   Experiments

We have evaluated our iterative spline fitting algorithm on the detection of line-features in aerial infra-red images. Figure 1a shows the original picture. The initial feature probabilities (i.e. $\pi(q_i^{(0)})$) assigned according to equation (1) are shown in Figure 1b. Figure 1c shows the final contour-map after the EM algorithm has converged. Notice that the line contours exhibit good connectivity and that the junctions are well reconstructed. We have highlighted a subregion of the original image. There are two features in this subregion to which we would like to draw attention. The first of these is centred on the junction structure. The second feature is a neighbouring point on the descending branch of the road.

Figure 2 shows the iterative evolution of the cubic splines at these two locations. The spline shown in Figure 2a adjusts to fit the upper pair of road segments. Notice also that although initially displaced, the final spline passes directly through the junction. In the case of the descending road-branch the spline shown in Figure 2b recovers from an initially poor orientation estimate to align itself with the underlying road structure. Figure 2c shows how the spline probabilities (i.e. $\pi(q_i^{(n)})$) evolve with iteration number. Initially, the neighbourhood is highly ambiguous. Many neighbouring splines compete to account for the local image structure. As a result the detected junction is several pixels wide. However, as the fitting process iterates, the splines move from the inconsistent initial estimate to give a good local estimate which is less ambiguous. In other words the two splines illustrated in Figure 2 have successfully arranged themselfs to account for the junction structure.

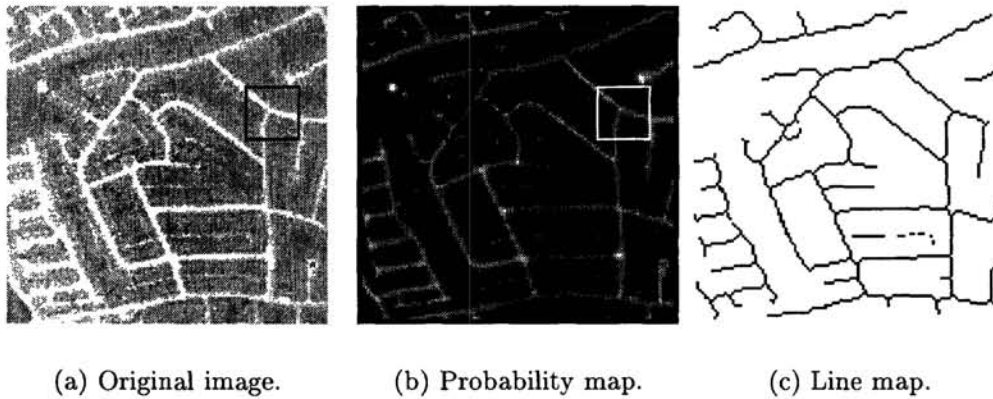

| (a) Original image. | (b) Probability map. | (c) Line map. |

Figure 1: Infra-red aerial picture with corresponding probability map showing region containing pixel under study and correspondent line map.

## 6 Conclusions

We have demonstrated how the process of parallel iterative contour refinement can be realised using the classical EM algorithm of Dempster, Laird and Rubin [1]. The refinement of curves by relaxation operations has been a preoccupation in the literature since the seminal work of Rosenfeld, Hummel and Zucker [9]. However, it is only recently that successful algorithms have been developed by appealing to more sophisticated modelling methodologies [13, 2]. Our EM approach not only delivers comparable performance, it does so using a very simple underlying model. Moreover, it allows the contour re-enforcement process to be understood in a weighted least-squares optimisation framework which has many features in common with snake dynamics [11] without being sensitive on the initial positioning of control points. Viewed from the perspective of classical relaxation labelling [9, 4], the EM framework provides a natural way of evaluating support beyond the immediate object neighbourhood. Moreover, the framework for spline fitting in 2D is readily extendible to the reconstruction of surface patches in 3D [10].

## References

[1] Dempster A., Laird N. and Rubin D., "Maximum-likelihood from incomplete data via the EM algorithm", J. Royal Statistical Soc. Ser. B (methodological),**39**, pp 1-38, 1977.

[2] Hancock E.R. and Kittler J., "Edge Labelling using Dictionary-based Probabilistic Relaxation", IEEE PAMI, **12**, pp. 161-185, 1990.

[3] Jordan M.I. and Jacobs R.A, "Hierarchical Mixtures of Experts and the EM Algorithm", *Neural Computation*, **6**, pp. 181-214, 1994.

[4] Kittler J. and Hancock, E.R., "Combining Evidence in Probabilistic relaxation", International Journal of Pattern Recognition and Artificial Intelligence, **3**, N1, pp 29-51, 1989.

[5] Leite J.A.F. and Hancock, E.R., " Statistically Combining and Refining Multichannel Information", *Progress in Image Analysis and Processing III: Edited by S Impedovo, World Scientific*, pp. 193-200, 1994.

[6] Leite J.A.F. and Hancock, E.R., "Iterative curve organisation with the EM algorithm", *to appear in Pattern Recognition Letters*, 1997.

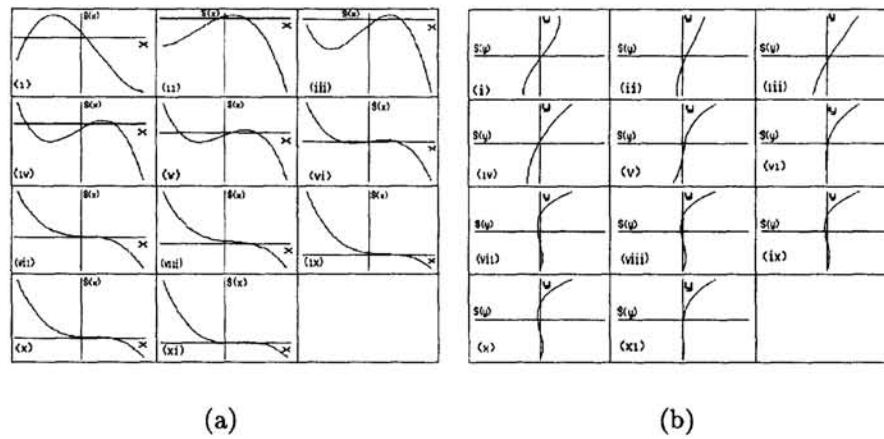

<div align="center">(a)                                                    (b)</div>

<div align="center">(c)</div>

Figure 2: Evolution of the spline in the fitting process. The image in (a) is the junction spline while the image in (b) is the branch spline. The first spline is shown in (i), and the subsequent ones from (ii) to (xi). The evolution of the corresponding spline probabilities is shown in (c).

[7] Meer P., Mintz D., Rosenfeld A. and Kim D.Y., "Robust Regression Methods for Computer Vision - A Review", *International Journal of Computer vision*, **6**, pp. 59–70, 1991.

[8] Peleg S. and Rosenfeld A., "Determining Compatibility Coefficients for curve enhancement relaxation processes", *IEEE SMC*, **8**, pp. 548–555, 1978.

[9] Rosenfeld A., Hummel R.A. and Zucker S.W., "Scene labelling by relaxation operations", IEEE Transactions SMC, SMC-6, pp400-433, 1976.

[10] Sander P.T. and Zucker S.W., "Inferring surface structure and differential structure from 3D images", IEEE PAMI, **12**, pp 833-854, 1990.

[11] Terzopoulos D., "Regularisation of inverse problems involving discontinuities", IEEE PAMI, **8**, pp 129-139, 1986.

[12] Zucker, S.W., Hummel R.A., and Rosenfeld A., "An application of relaxation labelling to line and curve enhancement", *IEEE TC*, **C-26**, pp. 394–403, 1977.

[13] Zucker S., David C., Dobbins A. and Iverson L., "The organisation of curve detection: coarse tangent fields and fine spline coverings", *Proceedings of the Second International Conference on Computer Vision*, pp. 577–586, 1988.
